# Combining Neural and Symbolic Learning to Revise Probabilistic Rule Bases

**J. Jeffrey Mahoney and Raymond J. Mooney**
Dept. of Computer Sciences
University of Texas
Austin, TX 78712
mahoney@cs.utexas.edu, mooney@cs.utexas.edu

## Abstract

This paper describes RAPTURE — a system for revising probabilistic knowledge bases that combines neural and symbolic learning methods. RAPTURE uses a modified version of backpropagation to refine the certainty factors of a MYCIN-style rule base and uses ID3's information gain heuristic to add new rules. Results on refining two actual expert knowledge bases demonstrate that this combined approach performs better than previous methods.

## 1  Introduction

In complex domains, learning needs to be biased with prior knowledge in order to produce satisfactory results from limited training data. Recently, both connectionist and symbolic methods have been developed for biasing learning with prior knowledge [Fu, 1989; Towell *et al.*, 1990; Ourston and Mooney, 1990]. Most of these methods revise an imperfect knowledge base (usually obtained from a domain expert) to fit a set of empirical data. Some of these methods have been successfully applied to real-world tasks, such as recognizing promoter sequences in DNA [Towell *et al.*, 1990; Ourston and Mooney, 1990]. The results demonstrate that revising an expert-given knowledge base produces more accurate results than learning from training data alone.

In this paper, we describe the RAPTURE system (Revising Approximate

Probabilistic Theories Using Repositories of Examples), which combines connectionist and symbolic methods to revise both the parameters and structure of a certainty-factor rule base.

## 2    The Rapture Algorithm

The RAPTURE algorithm breaks down into three main phases. First, an initial rule-base (created by a human expert) is converted into a RAPTURE network. The result is then trained using certainty-factor backpropagation (CFBP). The theory is further revised through network architecture modification. Once the network is fully trained, the solution is at hand—there is no need for retranslation. Each of these steps is outlined in full below.

### 2.1    The Initial Rule-Base

RAPTURE uses propositional certainty factor rules to represent its theories. These rules have the form $A \xrightarrow{0.8} D$, which expresses the idea that belief in proposition $A$ gives a 0.8 measure of belief in proposition $D$ [Shafer and Pearl, 1990]. Certainty factors can range in value from $-1$ to $+1$, and indicate a degree of confidence in a particular proposition. Certainty factor rules allow updating of these beliefs based upon new observed evidence.

Rules combine evidence via probabilistic sum, which is defined as $a \oplus b \equiv a + b - ab$. In general, all positive evidence is combined to determine the *measure of belief* (MB) for a given proposition, and all negative evidence is combined to obtain a *measure of disbelief* (MD). The certainty factor is then calculated using $CF = MB + MD$.

RAPTURE uses this formalism to represent its rule base for a variety of reasons. First, it is perhaps the simplest method that retains the desired evidence-summing aspect of uncertain reasoning. As each rule fires, additional evidence is contributed towards belief in the rule's consequent. The use of probabilistic sum enables many small pieces of evidence to add up to significant evidence. This is lacking in formalisms that use only MIN or MAX for combining evidence [Valtorta, 1988]. Second, probabilistic sum is a simple, differentiable, non-linear function. This is crucial for implementing gradient descent using backpropagation. Finally, and perhaps most significantly, is the widespread use of certainty factors. Numerous knowledge-bases have been implemented using this formalism, which immediately gives our approach a large base of applicability.

### 2.2    Converting the Rule Base into a Network

Once the initial theory is obtained, it is converted into a RAPTURE -network. Building the network begins by mapping all identical propositions in the rule-base to the same node in the network. Input features (those only appearing as rule-antecedents) become input nodes, and output symbols (those only appearing as rule-consequents) become output nodes. The certainty factors of the rules become the weights on the links that connect nodes. Networks for classification problems contain one output for each category. When an example is presented, the certainty factor for each of the categories is computed and the example is assigned to the category with the

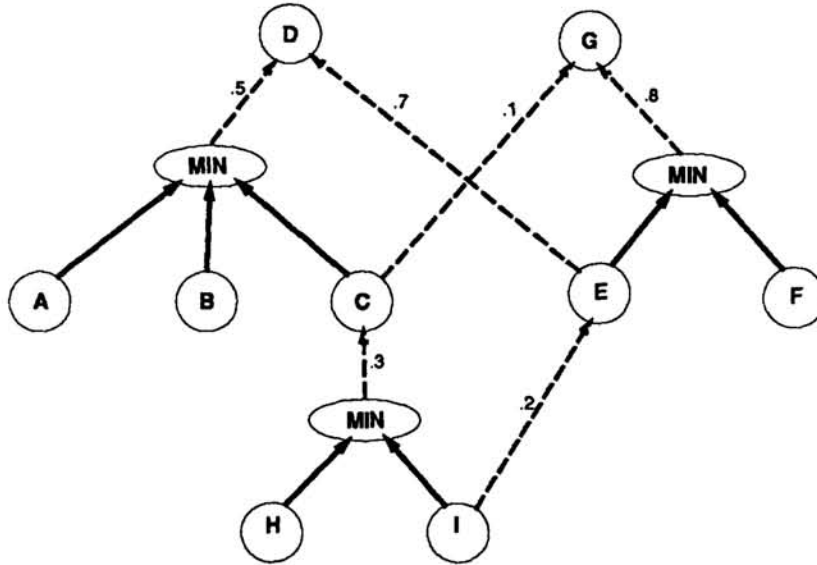

Figure 1: A RAPTURE NETWORK

highest value. Figure 1 illustrates the following set of rules.

$$ABC \xrightarrow{.5} D \quad E \xrightarrow{.7} D \quad C \xrightarrow{.1} G \quad EF \xrightarrow{.8} G \quad HI \xrightarrow{.3} C \quad I \xrightarrow{.2} E$$

As shown in the network, conjuncts must first pass through a MIN node before any activation reaches the consequent node. Note that each of the conjuncts is connected to the corresponding MIN mode with a solid line. This represents the fact that the link is non-adjustable, and simply passes its full activation value onto the MIN node. The standard (certainty-factor) links are drawn as dotted lines indicating that their values *are* adjustable.

This construction shows how easily a RAPTURE-network can model a MYCIN rule-base. Each representation can be converted into the other, without loss or corruption of information. They are two equivalent representations of the same set of rules.

### 2.3    Certainty Factor Backpropagation

Using the constructed RAPTURE-network, we desire to maximize its predictive accuracy over a set of training examples. Cycling through the examples one at a time, and slightly adjusting all relevant network weights in a direction that will minimize the output error, results in hill-climbing to a local minimum. This is the idea behind gradient descent [Rumelhart *et al.*, 1986], which RAPTURE accomplishes with Certainty Factor Backpropagation (CFBP), using the following equations.

$$\Delta_p w_{ji} = \eta \delta_{pj} (1 \pm \sum_{k \neq i} w_{jk} o_{pk}) \tag{1}$$

If $u_j$ is an output unit

$$\delta_{pj} = (t_{pj} - o_{pj}) \tag{2}$$

If $u_j$ is not an output unit

$$\delta_{pj} = \sum_{k_{min}} \delta_{pk} w_{kj} (1 \pm \sum_{i \neq k} w_{jk} o_{pk}) \tag{3}$$

The "Sigma with circle" notation is meant to represent probabilistic sum over the index, and the $\pm$ notation is shorthand for two separate cases. If $w_{ji} o_{pi} \geq 0$, then $-$ is used, otherwise $+$ is used. The $k_{min}$ subscript refers to the fact that we do not perform this summation for *every* unit $k$ (as in standard backpropagation), but only those units that received some contribution from unit $j$. Since a unit $j$ may be required to pass through a min or max-node before reaching the next layer $(k)$, it is possible that its value may not reach $k$.

RAPTURE deems a classification correct when the output value for the correct category is greater than that of any other category. No error propagation takes place in this case ($\delta_{pj} = 0$). CFBP terminates when overall error reaches a minimal value.

## 2.4   Changing the Network Architecture

Whenever training accuracy fails to reach 100% through CFBP, it may be an indication that the network architecture is inappropriate for the current classification task. To date, RAPTURE has been given two ways of changing network architecture. First, whenever the weight of a link in the network approaches zero, it is removed from the network along with all of the nodes and links that become detached due to this removal. Further, whenever an intermediate node loses all of its input links due to link deletion, it too is removed from the network, along with its output link. This link/node deletion is performed immediately after CFBP, and before anything new is introduced into the network.

RAPTURE also has a method for adding new nodes into the network. Specific nodes are added in an attempt to maximize the number of training examples that are classified correctly. The simple solution employed by RAPTURE is to create new input nodes that connect directly, either positively or negatively, to one or more output nodes. These new nodes are created in a way that will best help the network distinguish among training examples that are being misclassified. Specifically, RAPTURE attempts to distinguish for each output category, those examples of that category that are being misclassified (i.e. being classified into a different output category), from those examples that *do* belong in these different output categories. Quinlan's ID3 information gain metric [Quinlan, 1986] has been adopted by RAPTURE to select this new node, which becomes positive evidence for the correct category, and negative evidence for mistaken categories.

With these new nodes in place, we can now return to CFBP, where hopefully more training examples will be successfully classified. This entire process (CFBP followed by deleting links and adding new nodes) repeats until all training examples are correctly classified. Once this has occurred, the network is considered trained, and testing may begin.

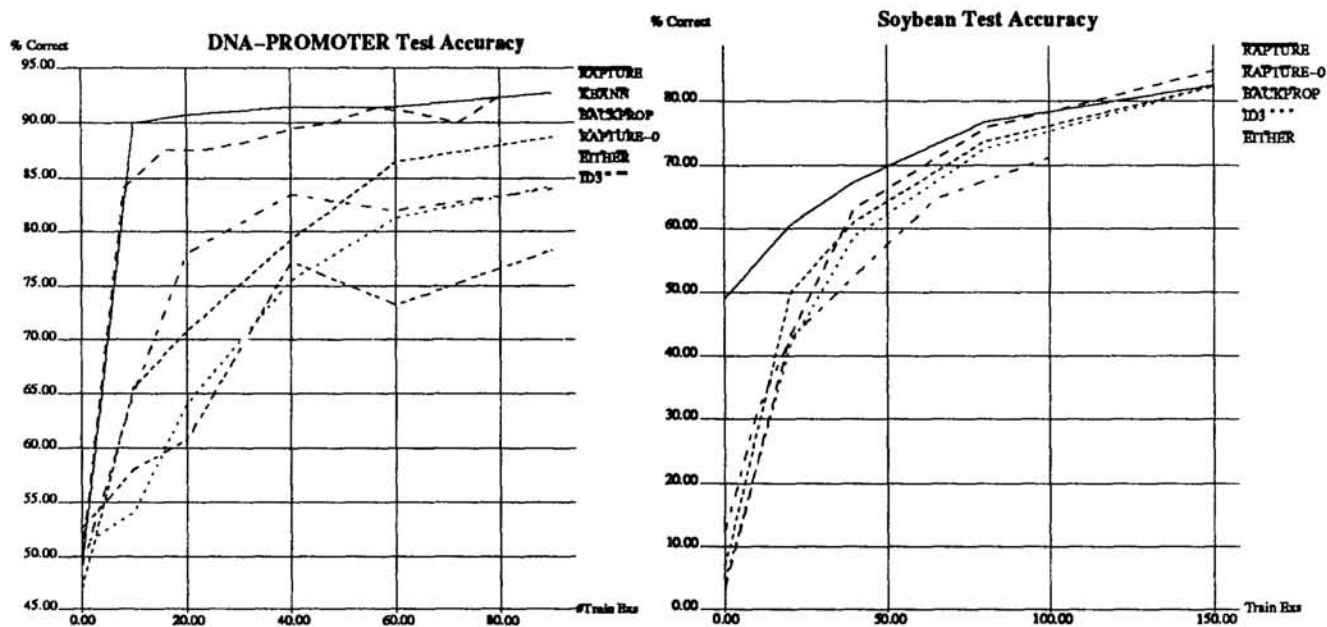

Figure 2: RAPTURE Testing Accuracy

## 3   Experimental Results

To date, RAPTURE has been tested on two real-world domains. The first of these is a domain for recognizing promoter sequences in strings of DNA-nucleotides. The second uses a theory for diagnosing soybean diseases. These datasets are discussed in detail in the following sections.

### 3.1   Promoter Recognition Results

A prokaryotic *promoter* is a short DNA sequence that precedes the beginnings of genes, and are locations where the protein RNA polymerase binds to the DNA structure [Towell *et al.*, 1990]. A set of propositional Horn-clause rules for recognizing promoters, along with 106 labelled examples (53 promoters, 53 non-promoters) was provided as the initial theory.

In order for this theory to used by RAPTURE it had to be modified into a certainty factor format. This was done by breaking up rules with multiple antecedents, into several rules. In this fashion, each antecedent is able to contribute some evidence towards belief in the consequent. Initial certainty factors were assigned in such a way that if *every* antecedent (from the original rule) were true, a certainty factor of 0.9 would result for the consequent.

To test RAPTURE using this dataset, standard training and test runs were performed, which resulted in the learning curve of Figure 2a. This graph is a plot of average performance in accuracy at classifying DNA strings over 25 independent trials. A single trial consists of providing each system with increasing numbers of examples to use for training, and then seeing how well it can classify unseen test examples. This graph clearly demonstrates the advantages of an evidence summing

system like RAPTURE over a pure Horn-clause system such as EITHER, a pure induc­tive system such as ID3, or a pure connectionist system, like backprop. Also plotted in the graph, is KBANN [Towell *et al.*, 1990], a symbolic-connectionist system that uses standard backpropagation, and RAPTURE-0, which is simply RAPTURE given no initial theory, emphasizing the importance of the expert knowledge. For this dataset, CFBP alone was all that was required in order to train the network. The node addition module was never called.

## 3.2  Soybean Disease Diagnosis Results

The Soybean Data comes from [Michalski and Chilausky, 1980] and is a dataset of 562 examples of diseased soybean plants. Examples were described by a string of 35 features including the condition of the stem, the roots, the seeds, as well as information such as the time of year, temperature, and features of the soil. An expert classified each example into one of 15 soybean diseases. This dataset has been used as a benchmark for a number of learning systems. Figure 2b is a learning curve on this data comparing RAPTURE, RAPTURE-0, backpropagation, ID3, and EITHER.

The headstart given to RAPTURE does not last throughout testing in this domain. RAPTURE maintains a statistically significant lead over the other systems (except RAPTURE-0) through 80 examples, but by 150 examples, all systems are performing at statistically equivalent levels. A likely explanation for this is that the expert provided theory is more helpful on the easier to diagnose diseases than on those that are more difficult. But these easy ones are also easy to learn via pure induction, and good rules can be created after seeing only a few examples. Trials have actually been run out to 300 examples, though all systems are performing at equivalent levels of accuracy.

# 4  Related Work

The SEEK system [Ginsberg *et al.*, 1988] revises rule bases containing *M-of-N* rules, though can not modify real-valued weights and contains no means for adding new rules. Valtorta [Valtorta, 1988] has examined the computational complexity of var­ious refinement tasks for probabilistic knowledge bases, and shows that refining the weights to fit a set of training data is an NP-Hard problem. Ma and Wilkins [Ma and Wilkins, 1991] have developed methods for improving the accuracy of a certainty-factor knowledge base by deleting rules, and they report modest improve­ments in the accuracy of a MYCIN rule base. Gallant [Gallant, 1988] designed and implemented a system that combines expert domain knowledge with connectionist learning, though is not suitable for multi-layer networks or for combination functions like probabilistic sum. KBANN [Towell *et al.*, 1990] uses standard backpropagation to refine a symbolic rule base, though the mapping between the symbolic rules and the network is only an approximation. Fu [Fu, 1989] and Lacher [Lacher, 1992] have also used backpropagation techniques to revise certainty factors on rules. However, the current publications on these two projects do not address the problem of alter­ing the network architecture (i.e. adding new rules) and do not present results on revising actual expert knowledge bases.

## 5  Future Work

The current method for changing network architecture in RAPTURE is restricted to adding new input units that directly feed the outputs. We hope to incorporate newer techniques for creating and linking to hidden nodes, in order to improve the range of architectural changes that it can make.

Another area requiring further research concerns the differences between certainty-factor networks and traditional connectionist networks. Further comparison of the RAPTURE and KBANN approaches to knowledge-base refinement are also indicated.

Finally, in recent years, certainty-factors have been the subject of considerable criticism from researchers in uncertain reasoning [Shafer and Pearl, 1990]. However, the basic revision framework in RAPTURE should be applicable to other uncertain reasoning formalisms such as Bayesian networks, Dempster-Shafer theory, or fuzzy logic [Shafer and Pearl, 1990]. As long as the activation functions in the corresponding network implementations of these methods are differentiable, backpropagation techniques should be employable.

## 6  Conclusions

Automatic refinement of probabilistic rule bases is an under-studied problem with important applications to the development of intelligent systems. This paper has described and evaluated an approach to refining certainty-factor rule bases that integrates connectionist and symbolic learning. The approach is implemented in a system called RAPTURE, which uses a revised backpropagation algorithm to modify certainty factors and ID3's information gain criteria to determine new rules to add to the network. In other words, connectionist methods are used to adjust parameters and symbolic methods are used to make structural changes to the knowledge base.

In domains with limited training data or domains requiring meaningful explanations for conclusions, refining existing expert knowledge has clear advantages. Results on revising three real-world knowledge bases indicates that RAPTURE generally performs better than purely inductive systems (ID3 and backpropagation), a purely symbolic revision system (EITHER), and and purely connectionist revision system (KBANN).

The certainty-factor networks used in RAPTURE blur the distinction between connectionist and symbolic representations. They can be viewed either as connectionist networks or symbolic rule bases. RAPTURE demonstrates the utility of applying connectionist learning methods to "symbolic" knowledge bases and employing symbolic methods to modify "connectionist" networks. Hopefully these results will encourage others to explore similar opportunities for cross-fertilization of ideas between connectionist and symbolic learning.

### Acknowledgements

This research was supported by the National Science Foundation under grant IRI-9102926, the NASA Ames Research Center under grant NCC 2-629, and the Texas Advanced Research Program under grant 003658114. We wish to thank R.S. Michal-

ski for furnishing the soybean data, M. Noordewier, G.G. Towell, and J.W. Shavlik for supplying the DNA data, and the KBANN results.

## References

[Fu, 1989] Li-Min Fu. Integration of neural heuristics into knowledge-based inference. *Connection Science*, 1(3):325–339, 1989.

[Gallant, 1988] S.I. Gallant. Connectionist expert systems. *Communications of the Association for Computing Machinery*, 31:152–169, 1988.

[Ginsberg *et al.*, 1988] A. Ginsberg, S. M. Weiss, and P. Politakis. Automatic knowledge based refinement for classification systems. *Artificial Intelligence*, 35:197–226, 1988.

[Lacher, 1992] R.C. Lacher. Expert networks: Paradigmatic conflict, technological rapprochement. Neuroprose FTP Archive, 1992.

[Ma and Wilkins, 1991] Y. Ma and D. C. Wilkins. Improving the performance of inconsistent knowledge bases via combined optimization method. In *Proceedings of the Eighth International Workshop on Machine Learning*, pages 23–27, Evanston, IL, June 1991.

[Michalski and Chilausky, 1980] R. S. Michalski and S. Chilausky. Learning by being told and learning from examples: An experimental comparison of the two methods of knowledge acquisition in the context of developing an expert system for soybean disease diagnosis. *Journal of Policy Analysis and Information Systems*, 4(2):126–161, 1980.

[Ourston and Mooney, 1990] D. Ourston and R. Mooney. Changing the rules: a comprehensive approach to theory refinement. In *Proceedings of the Eighth National Conference on Artificial Intelligence*, pages 815–820, Detroit, MI, July 1990.

[Quinlan, 1986] J. R. Quinlan. Induction of decision trees. *Machine Learning*, 1(1):81–106, 1986.

[Rumelhart *et al.*, 1986] D. E. Rumelhart, G. E. Hinton, and J. R. Williams. Learning internal representations by error propagation. In D. E. Rumelhart and J. L. McClelland, editors, *Parallel Distributed Processing, Vol. I*, pages 318–362. MIT Press, Cambridge, MA, 1986.

[Shafer and Pearl, 1990] G. Shafer and J. Pearl, editors. *Readings in Uncertain Reasoning*. Morgan Kaufmann, Inc., San Mateo,CA, 1990.

[Towell *et al.*, 1990] G. G. Towell, J. W. Shavlik, and Michiel O. Noordewier. Refinement of approximate domain theories by knowledge-based artificial neural networks. In *Proceedings of the Eighth National Conference on Artificial Intelligence*, pages 861–866, Boston, MA, July 1990.

[Valtorta, 1988] M. Valtorta. Some results on the complexity of knowledge-base refinement. In *Proceedings of the Sixth International Workshop on Machine Learning*, pages 326–331, Ithaca, NY, June 1988.